# Classifying Facial Action

**Marian Stewart Bartlett, Paul A. Viola,**
**Terrence J. Sejnowski, Beatrice A. Golomb**
Howard Hughes Medical Institute
The Salk Institute, La Jolla, CA 92037
marni, viola, terry, beatrice @salk.edu

**Jan Larsen**
The Niels Bohr Institute
2100 Copenhagen
Denmark
jlarsen@fys.ku.dk

**Joseph C. Hager**
Network Information Research Corp
Salt Lake City, Utah
jchager@ibm.net

**Paul Ekman**
University of California San Francisco
San Francisco, CA 94143
ekmansf@itsa.ucsf.edu

## Abstract

The Facial Action Coding System, (FACS), devised by Ekman and
Friesen (1978), provides an objective means for measuring the facial
muscle contractions involved in a facial expression. In this paper,
we approach automated facial expression analysis by detecting and
classifying facial actions. We generated a database of over 1100
image sequences of 24 subjects performing over 150 distinct facial
actions or action combinations. We compare three different ap-
proaches to classifying the facial actions in these images: Holistic
spatial analysis based on principal components of graylevel images;
explicit measurement of local image features such as wrinkles; and
template matching with motion flow fields. On a dataset contain-
ing six individual actions and 20 subjects, these methods had 89%,
57%, and 85% performances respectively for generalization to novel
subjects. When combined, performance improved to 92%.

## 1 INTRODUCTION

Measurement of facial expressions is important for research and assessment psychi-
atry, neurology, and experimental psychology (Ekman, Huang, Sejnowski, & Hager,
1992), and has technological applications in consumer-friendly user interfaces, inter-
active video and entertainment rating. The Facial Action Coding System (FACS)
is a method for measuring facial expressions in terms of activity in the underlying
facial muscles (Ekman & Friesen, 1978). We are exploring ways to automate FACS.

Rather than classifying images into emotion categories such as happy, sad, or surprised, the goal of this work is instead to detect the muscular actions that comprise a facial expression.

FACS was developed in order to allow researchers to measure the activity of facial muscles from video images of faces. Ekman and Friesen defined 46 distinct action units, each of which correspond to activity in a distinct muscle or muscle group, and produce characteristic facial distortions which can be identified in the images. Although there are static cues to the facial actions, dynamic information is a critical aspect of facial action coding.

FACS is currently used as a research tool in several branches of behavioral science, but a major limitation to this system is the time required to both train human experts and to manually score the video tape. Automating the Facial Action Coding System would make it more widely accessible as a research tool, and it would provide a good foundation for human-computer interactions tools.

**Why Detect Facial Actions?**

Most approaches to facial expression recognition by computer have focused on classifying images into a small set of emotion categories such as happy, sad, or surprised (Mase, 1991; Yacoob & Davis, 1994; Essa & Pentland, 1995). Real facial signals, however, consist of thousands of distinct expressions, that differ often in only subtle ways. These differences can signify not only which emotion is occurring, but whether two or more emotions have blended together, the intensity of the emotion(s), and if an attempt is being made to control the expression of emotion (Hager & Ekman, 1995).

An alternative to training a system explicitly on a large number of expression categories is to detect the facial actions that comprise the expressions. Thousands of facial expressions can be defined in terms of this smaller set of structural components. We can verify the signal value of these expressions by reference to a large body of behavioral data relating facial actions to emotional states which have already been scored with FACS. FACS also provides a means for obtaining reliable training data. Other approaches to automating facial measurement have mistakenly relied upon voluntary expressions, which tend to contain exaggerated and redundant cues, while omitting some muscular actions altogether (Hager & Ekman, 1995).

## 2  IMAGE DATABASE

We have collected a database of image sequences of subjects performing specified facial actions. The full database contains over 1100 sequences containing over 150 distinct actions, or action combinations, and 24 different subjects. The sequences contain 6 images, beginning with a neutral expression and ending with a high intensity muscle contraction (Figure 1). For our initial investigation we used data from 20 subjects and attempted to classify the six individual upper face actions illustrated in Figure 2. The information that is available in the images for detecting and discriminating these actions include distortions in the shapes and relative positions of the eyes and eyebrows, the appearance of wrinkles, bulges, and furrows, in specific regions of the face, and motion of the brows and eyelids.

Prior to classifying the images, we manually located the eyes, and we used this information to crop a region around the upper face and scale the images to 360 x 240. The images were rotated so that the eyes were horizontal, and the luminance was normalized. Accurate image registration is critical for principal components based approaches. For the holistic analysis and flow fields, the images were further scaled

to 22 x 32 and 66 x 96, respectively. Since the muscle contractions are frequently asymmetric about the face, we doubled the size of our data set by reflecting each image about the vertical axis, giving a total of 800 images.

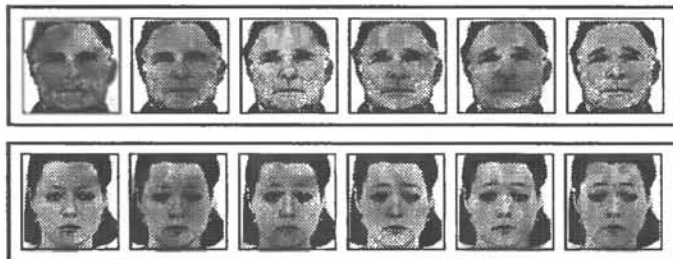

Figure 1: Example action sequences from the database.

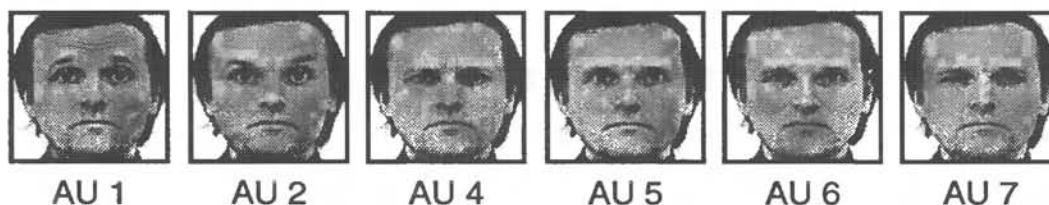

AU 1          AU 2          AU 4          AU 5          AU 6          AU 7

Figure 2: Examples of the six actions used in this study. AU 1: Inner brow raiser. 2: Outer brow raiser. 4: Brow lower. 5: Upper lid raiser (widening the eyes). 6: Cheek raiser. 7: Lid tightener (partial squint).

## 3   HOLISTIC SPATIAL ANALYSIS

The Eigenface (Turk & Pentland, 1991) and Holon (Cottrell & Metcalfe, 1991) representations are holistic representations based on principal components, which can be extracted by feed forward networks trained by back propagation. Previous work in our lab and others has demonstrated that feed forward networks taking such holistic representations as input can successfully classify gender from facial images (Cottrell & Metcalfe, 1991; Golomb, Lawrence, & Sejnowski, 1991). We evaluated the ability of a back propagation network to classify facial actions given principal components of graylevel images as input.

The primary difference between the present approach and the work referenced above is that we take the principal components of a set of difference images, which we obtained by subtracting the first image in the sequence from the subsequent images (see Figure 3). The variability in our data set is therefore due to the facial distortions and individual differences in facial distortion, and we have removed variability due to surface-level differences in appearance.

We projected the difference images onto the first N principal components of the dataset, and these projections comprised the input to a 3 layer neural network with 10 hidden units, and six output units, one per action (Figure 3.) The network is feed forward and fully connected with a hyperbolic tangent transfer function, and was trained with conjugate gradient descent. The output of the network was determined using winner take all, and generalization to novel subjects was determined by using the leave-one-out, or jackknife, procedure in which we trained the network on 19 subjects and reserved all of the images from one subject for testing. This process was repeated for each of the subjects to obtain a mean generalization performance across 20 test cases.

We obtained the best performance with 50 component projections, which gave 88.6% correct across subjects. The benefit obtained by using principal components over the 704-dimensional difference images themselves is not large. Feeding the difference images directly into the network gave a performance of 84% correct.

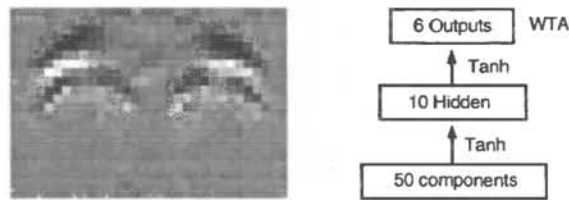

Figure 3: Left: Example difference image. Input values of -1 are mapped to black and 1 to white. Right: Architecture of the feed forward network.

## 4  FEATURE MEASUREMENT

We turned next to explicit measurement of local image features associated with these actions. The presence of wrinkles in specific regions of the face is a salient cue to the contraction of specific facial muscles. We measured wrinkling at the four facial positions marked in Figure 4a, which are located in the image automatically from the eye position information. Figure 4b shows pixel intensities along the line segment labeled A, and two major wrinkles are evident.

We defined a wrinkle measure P as the sum of the squared derivative of the intensity values along the segment (Figure 4c.) Figure 4d shows P values along line segment A, for a subject performing each of the six actions. Only AU 1 produces wrinkles in the center of the forehead. The P values remain at zero except for AU 1, for which it increases with increases in action intensity. We also defined an eye opening measure as the area of the visible sclera lateral to the iris. Since we were interested in changes in these measures from baseline, we subtract the measures obtained from the neutral image.

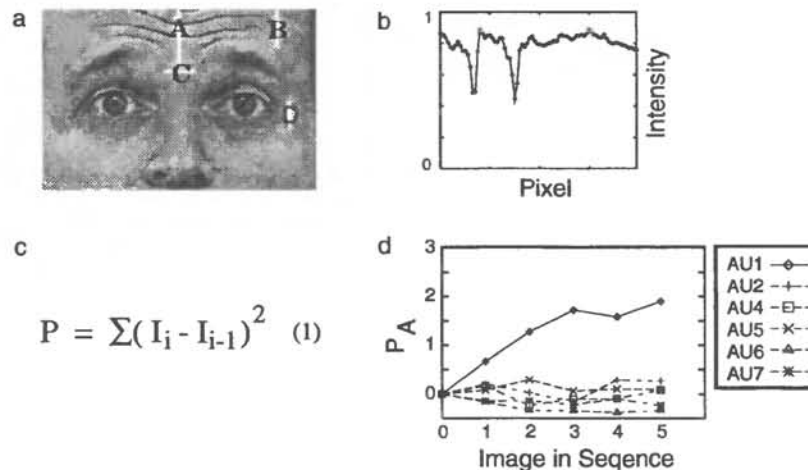

Figure 4: a) Wrinkling was measured at four image locations, A-D. b) Smoothed pixel intensities along the line labeled A. c) Wrinkle measure. d) P measured at image location A for one subject performing each of the six actions.

We classified the actions from these five feature measures using a 3-layer neural net with 15 hidden units. This method performs well for some subjects but not for

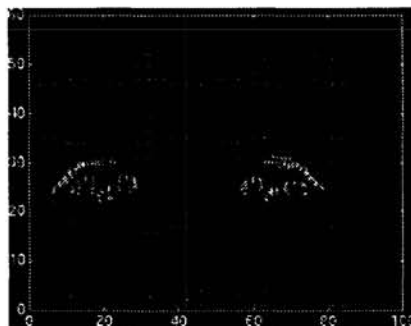

Figure 5: Example flow field for a subject performing AU 7, partial closure of the eyelids. Each flow vector is plotted as an arrow that points in the direction of motion. Axes give image location.

others, depending on age and physiognomy. It achieves an overall generalization performance of 57% correct.

## 5 OPTIC FLOW

The motion that results from facial action provides another important source of information. The third classifier attempts to classify facial actions based only on the pattern of facial motion. Motion is extracted from image pairs consisting of a neutral image and an image that displays the action to be classified. An approximation to flow is extracted by implementing the brightness constraint equation (2) where the velocity $(v_x, v_y)$ at each image point is estimated from the spatial and temporal gradients of the image $I$. The velocities can only be reliably extracted at points of large gradient, and we therefore retain only the velocities from those locations. One of the advantages of this simple local estimate of flow is speed. It takes 0.13 seconds on a 120 MHz Pentium to compute one flow field. A resulting flow image is illustrated in Figure 5.

$$v_x \frac{\partial I(x,y,t)}{\partial x} + v_y \frac{\partial I(x,y,t)}{\partial y} + \frac{\partial I(x,y,t)}{\partial t} = 0 \qquad (2)$$

We obtained weighted templates for each of the actions by taking mean flow fields from 10 subjects. We compared novel flow patterns, $f^n$ to the template $f^t$ by the similarity measure S (3). S is the normalized dot product of the novel flow field with the template flow field. This template matching procedure gave 84.8% accuracy for novel subjects. Performance was the same for the ten subjects used in the training set as for the ten in the test set.

$$S(f^n, f^t) = \frac{\sum_i f_i^n \cdot f_i^t}{\sqrt{\sum_i f_i^n \cdot f_i^n} \sqrt{\sum_i f_i^t \cdot f_i^t}} \qquad (3)$$

## 6 COMBINED SYSTEM

Figure 6 compares performance for the three individual methods described in the previous sections. Error bars give the standard deviation for the estimate of generalization to novel subjects. We obtained the best performance when we combined all three sources of information into a single neural network. The classifier is a

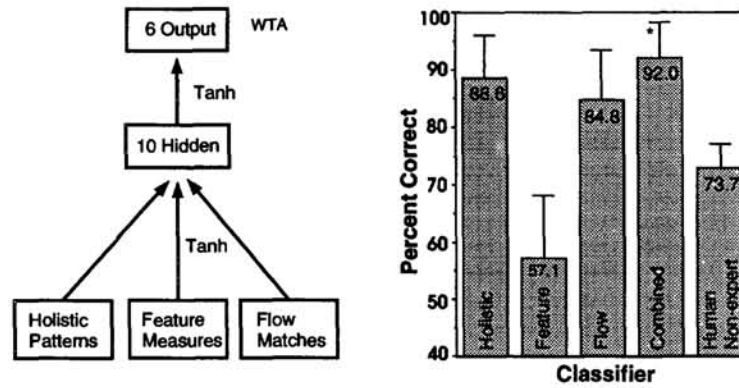

Figure 6: Left: Combined system architecture. Right: Performance comparisons.

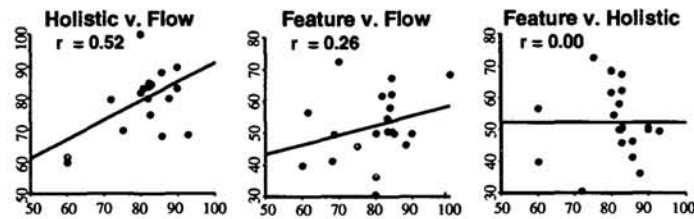

Figure 7: Performance correlations among the three individual classifiers. Each data point is performance for one of the 20 subjects.

feed forward network taking 50 component projections, 5 feature measures, and 6 template matches as input (see Figure 6.)

The combined system gives a generalization performance of 92%, which is an improvement over the best individual method at 88.6%. The increase in performance level is statistically significant by a paired t-test. While the improvement is small, it constitutes about 30% of the difference between the best individual classifier and perfect performance. Figure 6 also shows performance of human subjects on this same dataset. Human non-experts can correctly classify these images with about 74% accuracy. This is a difficult classification problem that requires considerable training for people to be able to perform well.

We can examine how the combined system benefits from multiple input sources by looking at the correlations in performance of the three individual classifiers. Combining estimators is most beneficial when the individual estimators make very different patterns of errors.[1] The performance of the individual classifiers are compared in Figure 7.

The holistic and the flow field classifiers are correlated with a coefficient of 0.52. The feature based system, however, has a more independent pattern of errors from the two template-based methods. Although the stand-alone performance of the feature-based system is low, it contributes to the combined system because it provides estimates that are independent from the two template-based systems. Without the feature measures, we lose 40% of the improvement. Since we have only a small number of features, this data does not address questions about whether templates are *better* than features, but it does suggest that local features plus templates may be superior to either one alone, since they may have independent patterns of errors.

# 7 DISCUSSION

We have evaluated the performance of three approaches to image analysis on a difficult classification problem. We obtained the best performance when information from holistic spatial analysis, feature measurements, and optic flow fields were combined in a single system. The combined system classifies a face in less than a second on a 120 MHz Pentium.

Our initial results are promising since the upper facial actions included in this study represent subtle distinctions in facial appearance that require lengthy training for humans to make reliably. Our results compare favorably with facial expression recognition systems developed by Mase (1991), Yacoob and Davis (1994), and Padgett and Cottrell (1995), who obtained 80%, 88%, and 88% accuracy respectively for classifying up to six full face expressions. The work presented here differs from these systems in that we attempt to detect individual muscular actions rather than emotion categories, we use a dataset of labeled facial actions, and our dataset includes low and medium intensity muscular actions as well as high intensity ones. Essa and Pentland (1995) attempt to relate facial expressions to the underlying musculature through a complex physical model of the face. Since our methods are image-based, they are more adaptable to variations in facial structure and skin elasticity in the subject population.

We intend to apply these techniques to the lower facial actions and to action combinations as well. A completely automated method for scoring facial actions from images would have both commercial and research applications and would reduce the time and expense currently required for manual scoring by trained observers.

## Acknowledgments

This research was supported by Lawrence Livermore National Laboratories, Intra-University Agreement B291436, NSF Grant No. BS-9120868, and Howard Hughes Medical Institute. We thank Claudia Hilburn for image collection.

## Footnotes

[1]Tom Dietterich, Connectionists mailing list, July 24, 1993.

## References

Cottrell, G.,& Metcalfe, J. (1991): Face, gender and emotion recognition using holons. In *Advances in Neural Information Processing Systems 3*, D. Touretzky, (Ed.) San Mateo: Morgan & Kaufman. 564 - 571.

Ekman, P., & Friesen, W. (1978): Facial Action Coding System: A Technique for the Measurement of Facial Movement. Palo Alto, CA: *Consulting Psychologists Press*.

Ekman, P., Huang, T., Sejnowski, T., & Hager, J. (1992): Final Report to NSF of the Planning Workshop on Facial Expression Understanding. Available from HIL-0984, UCSF, San Francisco, CA 94143.

Essa, I., & Pentland, A. (1995). Facial expression recognition using visually extracted facial action parameters. *Proceedings of the International Workshop on Automatic Face- and Gesture-Recognition*. University of Zurich, Multimedia Laboratory.

Golomb, B., Lawrence, D., & Sejnowski, T. (1991). SEXnet: A neural network identifies sex from human faces. In *Advances in Neural Information Processing Systems 3*, D. Touretzky, (Ed.) San Mateo: Morgan & Kaufman: 572 - 577.

Hager, J., & Ekman, P., (1995). The essential behavioral science of the face and gesture that computer scientists need to know. *Proceedings of the International Workshop on Automatic Face- and Gesture-Recognition*. University of Zurich, Multimedia Laboratory.

Mase, K. (1991): Recognition of facial expression from optical flow. *IEICE Transactions E* 74(10): 3474-3483.

Padgett, C., Cottrell, G., (1995). Emotion in static face images. *Proceedings of the Institute for Neural Computation Annual Research Symposium, Vol 5*. La Jolla, CA.

Turk, M., & Pentland, A. (1991): Eigenfaces for Recognition. *Journal of Cognitive Neuroscience* 3(1): 71 - 86.

Yacoob, Y., & Davis, L. (1994): Recognizing human facial expression. *University of Maryland Center for Automation Research Technical Report No. 706.*